# 3D Object Recognition with Deep Belief Nets

**Vinod Nair and Geoffrey E. Hinton**
Department of Computer Science, University of Toronto
10 King's College Road, Toronto, M5S 3G5 Canada
{vnair,hinton}@cs.toronto.edu

## Abstract

We introduce a new type of top-level model for Deep Belief Nets and evaluate it on a 3D object recognition task. The top-level model is a third-order Boltzmann machine, trained using a hybrid algorithm that combines both generative and discriminative gradients. Performance is evaluated on the NORB database (*normalized-uniform* version), which contains stereo-pair images of objects under different lighting conditions and viewpoints. Our model achieves 6.5% error on the test set, which is close to the best published result for NORB (5.9%) using a convolutional neural net that has built-in knowledge of translation invariance. It substantially outperforms shallow models such as SVMs (11.6%). DBNs are especially suited for semi-supervised learning, and to demonstrate this we consider a modified version of the NORB recognition task in which additional unlabeled images are created by applying small translations to the images in the database. With the extra unlabeled data (and the same amount of labeled data as before), our model achieves 5.2% error.

## 1   Introduction

Recent work on deep belief nets (DBNs) [10], [13] has shown that it is possible to learn multiple layers of non-linear features that are useful for object classification without requiring labeled data. The features are trained one layer at a time as a restricted Boltzmann machine (RBM) using contrastive divergence (CD) [4], or as some form of autoencoder [20], [16], and the feature activations learned by one module become the data for training the next module. After a pre-training phase that learns layers of features which are good at modeling the statistical structure in a set of unlabeled images, supervised backpropagation can be used to fine-tune the features for classification [7]. Alternatively, classification can be performed by learning a top layer of features that models the joint density of the class labels and the highest layer of unsupervised features [6]. These unsupervised features (plus the class labels) then become the penultimate layer of the deep belief net [6].

Early work on deep belief nets was evaluated using the MNIST dataset of handwritten digits [6] which has the advantage that a few million parameters are adequate for modeling most of the structure in the domain. For 3D object classification, however, many more parameters are probably required to allow a deep belief net with no prior knowledge of spatial structure to capture all of the variations caused by lighting and viewpoint. It is not yet clear how well deep belief nets perform at 3D object classification when compared with shallow techniques such as SVM's [19], [3] or deep discriminative techniques like convolutional neural networks [11].

In this paper, we describe a better type of top-level model for deep belief nets that is trained using a combination of generative and discriminative gradients [5], [8], [9]. We evaluate the model on NORB [12], which is a carefully designed object recognition task that requires

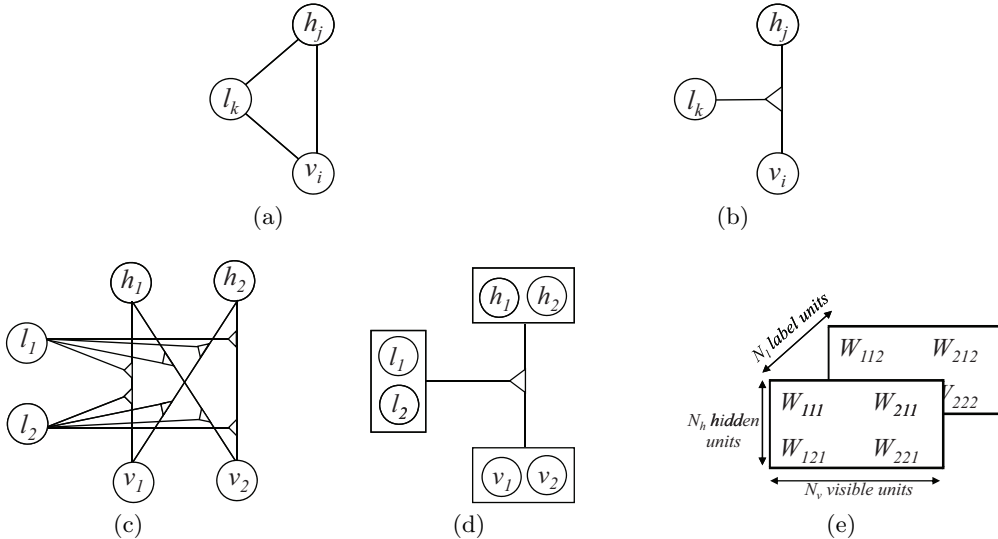

Figure 1: The Third-Order Restricted Boltzmann Machine. **(a)** Every clique in the model contains a visible unit, hidden unit, and label unit. **(b)** Our shorthand notation for representing the clique in (a). **(c)** A model with two of each unit type. There is one clique for every possible triplet of units created by selecting one of each type. The "restricted" architecture precludes cliques with multiple units of the same type. **(d)** Our shorthand notation for representing the model in (c). **(e)** The 3D tensor of parameters for the model in (c). The architecture is the same as that of an implicit mixture of RBMs [14], but the inference and learning algorithms have changed.

generalization to novel object instances under varying lighting conditions and viewpoints. Our model significantly outperforms SVM's, and it also outperforms convolutional neural nets when given additional *unlabeled* data produced by small translations of the training images. We use restricted Boltzmann machines trained with one-step contrastive divergence as our basic module for learning layers of features. These are fully described elsewhere [6], [1] and the reader is referred to those sources for details.

## 2   A Third-Order RBM as the Top-Level Model

Until now, the only top-level model that has been considered for a DBN is an RBM with two types of observed units (one for the label, another for the penultimate feature vector). We now consider an alternative model for the top-level joint distribution in which the class label multiplicatively interacts with *both* the penultimate layer units and the hidden units to determine the energy of a full configuration. It is a Boltzmann machine with three-way cliques [17], each containing a penultimate layer unit $v_i$, a hidden unit $h_j$, and a label unit $l_k$. See figure 1 for a summary of the architecture. Note that the parameters now form a *3D tensor*, instead of a matrix as in the earlier, bipartite model.

Consider the case where the components of $\mathbf{v}$ and $\mathbf{h}$ are stochastic binary units, and $\mathbf{l}$ is a discrete variable with $K$ states represented by 1-of-$K$ encoding. The model can be defined in terms of its energy function

$$E(\mathbf{v}, \mathbf{h}, \mathbf{l}) = -\sum_{i,j,k} W_{ijk} v_i h_j l_k, \tag{1}$$

where $W_{ijk}$ is a learnable scalar parameter. (We omit bias terms from all expressions for clarity.) The probability of a full configuration $\{\mathbf{v}, \mathbf{h}, \mathbf{l}\}$ is then

$$P(\mathbf{v}, \mathbf{h}, \mathbf{l}) = \frac{\exp(-E(\mathbf{v}, \mathbf{h}, \mathbf{l}))}{Z}, \tag{2}$$

where $Z = \sum_{\mathbf{v}', \mathbf{h}', \mathbf{l}'} \exp(-E(\mathbf{v}', \mathbf{h}', \mathbf{l}'))$ is the partition function. Marginalizing over $\mathbf{h}$ gives the distribution over $\mathbf{v}$ and $\mathbf{l}$ alone.

The main difference between the new top-level model and the earlier one is that now the class label multiplicatively modulates how the visible and hidden units contribute to the energy of a full configuration. If the label's $k^{th}$ unit is 1 (and the rest are 0), then the $k^{th}$ slice of the tensor determines the energy function. In the case of soft activations (i.e. more than one label has non-zero probability), a weighted blend of the tensor's slices specifies the energy function. The earlier top-level (RBM) model limits the label's effect to changing the biases into the hidden units, which modifies only how the hidden units contribute to the energy of a full configuration. There is no direct interaction between the label and the visible units. Introducing direct interactions among all three sets of variables allows the model to learn features that are dedicated to each class. This is a useful property when the object classes have substantially different appearances that require very different features to describe. Unlike an RBM, the model structure is not bipartite, but it is still "restricted" in the sense that there are no direct connections between two units of the same type.

## 2.1 Inference

The distributions that we would like to be able to infer are $P(\mathbf{l}|\mathbf{v})$ (to classify an input), and $P(\mathbf{v},\mathbf{l}|\mathbf{h})$ and $P(\mathbf{h}|\mathbf{v},\mathbf{l})$ (for CD learning). Fortunately, all three distributions are tractable to sample from exactly. The simplest case is $P(\mathbf{h}|\mathbf{v},\mathbf{l})$. Once $\mathbf{l}$ is observed, the model reduces to an RBM whose parameters are the $k^{th}$ slice of the 3D parameter tensor. As a result $P(\mathbf{h}|\mathbf{v},\mathbf{l})$ is a factorized distribution that can be sampled exactly.

For a restricted third-order model with $N_v$ visible units, $N_h$ hidden units and $N_l$ class labels, the distribution $P(\mathbf{l}|\mathbf{v})$ can be exactly computed in $O(N_v N_h N_l)$ time. This result follows from two observations: 1) setting $l_k = 1$ reduces the model to an RBM defined by the $k^{th}$ slice of the tensor, and 2) the negative log probability of $\mathbf{v}$, up to an additive constant, under this RBM is the *free energy*:

$$F_k(\mathbf{v}) = -\sum_{j=1}^{N_h} \log(1 + \exp(\sum_{i=1}^{N_v} W_{ijk} v_i)). \tag{3}$$

The idea is to first compute $F_k(\mathbf{v})$ for each setting of the label, and then convert them to a discrete distribution by taking the *softmax* of the negative free energies:

$$P(l_k = 1|\mathbf{v}) = \frac{\exp(-F_k(\mathbf{v}))}{\sum_{k=1}^{N_l} \exp(-F_k(\mathbf{v}))}. \tag{4}$$

Equation 3 requires $O(N_v N_h)$ computation, which is repeated $N_l$ times for a total of $O(N_v N_h N_l)$ computation.

We can use the same method to compute $P(\mathbf{l}|\mathbf{h})$. Simply switch the role of $\mathbf{v}$ and $\mathbf{h}$ in equation 3 to compute the free energy of $\mathbf{h}$ under the $k^{th}$ RBM. (This is possible since the model is symmetric with respect to $\mathbf{v}$ and $\mathbf{h}$.) Then convert the resulting $N_l$ free energies to the probabilities $P(l_k = 1|\mathbf{h})$ with the softmax function.

Now it becomes possible to exactly sample $P(\mathbf{v},\mathbf{l}|\mathbf{h})$ by first sampling $\tilde{\mathbf{l}} \sim P(\mathbf{l}|\mathbf{h})$. Suppose $\tilde{l}_k = 1$. Then the model reduces to its $k^{th}$-slice RBM from which $\tilde{\mathbf{v}} \sim P(\mathbf{v}|\mathbf{h}, \tilde{l}_k = 1)$ can be easily sampled. The final result $\{\tilde{\mathbf{v}}, \tilde{\mathbf{l}}\}$ is an unbiased sample from $P(\mathbf{v},\mathbf{l}|\mathbf{h})$.

## 2.2 Learning

Given a set of $N$ labeled training cases $\{(\mathbf{v}_1, l_1), (\mathbf{v}_2, \mathbf{l}_2), ..., (\mathbf{v}_N, \mathbf{l}_N)\}$ , we want to learn the 3D parameter tensor $W$ for the restricted third-order model. When trained as the top-level model of a DBN, the visible vector $\mathbf{v}$ is a penultimate layer feature vector. We can also train the model directly on images as a shallow model, in which case $\mathbf{v}$ is an image (in row vector form). In both cases the label $\mathbf{l}$ represents the $N_l$ object categories using 1-of-$N_l$ encoding. For the same reasons as in the case of an RBM, maximum likelihood learning is intractable here as well, so we rely on Contrastive Divergence learning instead. CD was originally formulated in the context of the RBM and its bipartite architecture, but here we extend it to the non-bipartite architecture of the third-order model.

An unbiased estimate of the maximum likelihood gradient can be computed by running a Markov chain that alternatively samples $P(\mathbf{h}|\mathbf{v},\mathbf{l})$ and $P(\mathbf{v},\mathbf{l}|\mathbf{h})$ until it reaches equilibrium. Contrastive divergence uses the parameter updates given by three half-steps of this chain, with the chain initialized from a training case (rather than a random state). As explained in section 2.1, both of these distributions are easy to sample from. The steps for computing the CD parameter updates are summarized below:

Contrastive divergence learning of $P(\mathbf{v},\mathbf{l})$:

1. Given a labeled training pair $\{\mathbf{v}^+, l_k^+ = 1\}$, sample $\mathbf{h}^+ \sim P(\mathbf{h}|\mathbf{v}^+, l_k^+ = 1)$.
2. Compute the outer product $D_k^+ = \mathbf{v}^+ \mathbf{h}^{+T}$.
3. Sample $\{\mathbf{v}^-, \mathbf{l}^-\} \sim P(\mathbf{v},\mathbf{l}|\mathbf{h}^+)$. Let $m$ be the index of the component of $\mathbf{l}^-$ set to 1.
4. Sample $\mathbf{h}^- \sim P(\mathbf{h}|\mathbf{v}^-, l_m^- = 1)$.
5. Compute the outer product $D_m^- = \mathbf{v}^- \mathbf{h}^{-T}$.

Let $W_{\cdot,\cdot,k}$ denote the $N_h \times N_v$ matrix of parameters corresponding to the $k^{th}$ slice along the label dimension of the 3D tensor. Then the CD update for $W_{\cdot,\cdot,k}$ is:

$$\Delta W_{\cdot,\cdot,k} = D_k^+ - D_k^-, \tag{5}$$

$$W_{\cdot,\cdot,k} \leftarrow W_{\cdot,\cdot,k} + \eta \Delta W_{\cdot,\cdot,k}, \tag{6}$$

where $\eta$ is a learning rate parameter. Typically, the updates computed from a "mini-batch" of training cases (a small subset of the entire training set) are averaged together into one update and then applied to the parameters.

## 3 Combining Gradients for Generative and Discriminative Models

In practice the Markov chain used in the learning of $P(\mathbf{v},\mathbf{l})$ can suffer from slow mixing. In particular, the label $\mathbf{l}^-$ generated in step 3 above is unlikely to be different from the true label $\mathbf{l}^+$ of the training case used in step 1. Empirically, the chain has a tendency to stay "stuck" on the same state for the label variable because in the positive phase the hidden activities are inferred with the label clamped to its true value. So the hidden activities contain information about the true label, which gives it an advantage over the other labels.

Consider the extreme case where we initialize the Markov chain with a training pair $\{\mathbf{v}^+, l_k^+ = 1\}$ and the label variable *never* changes from its initial state during the chain's entire run. In effect, the model that ends up being learned is a class-conditional generative distribution $P(\mathbf{v}|l_k = 1)$, represented by the $k^{th}$ slice RBM. The parameter updates are identical to those for training $N_l$ independent RBMs, one per class, with only the training cases of each class being used to learn the RBM for that class. Note that this is very different from the model in section 2: here the energy functions implemented by the class-conditional RBMs are learned independently and their energy units are not commensurate with each other.

Alternatively, we can optimize the *same* set of parameters to represent yet another distribution, $P(\mathbf{l}|\mathbf{v})$. The advantage in this case is that the *exact* gradient needed for maximum likelihood learning, $\partial log P(\mathbf{l}|\mathbf{v})/\partial W$, can be computed in $O(N_v N_h N_l)$ time. The gradient expression can be derived with some straightforward differentiation of equation 4. The disadvantage is that it cannot make use of unlabeled data. Also, as the results show, learning a purely discriminative model at the top level of a DBN gives much worse performance.

However, now a new way of learning $P(\mathbf{v},\mathbf{l})$ becomes apparent: we can optimize the parameters by using *a weighted sum of the gradients* for $\log P(\mathbf{v}|\mathbf{l})$ and $\log P(\mathbf{l}|\mathbf{v})$. As explained below, this approach 1) avoids the slow mixing of the CD learning for $P(\mathbf{v},\mathbf{l})$, and 2) allows learning with both labeled and unlabeled data. It resembles pseudo-likelihood in how it optimizes the two conditional distributions in place of the joint distribution, except here one of the conditionals ($P(\mathbf{v}|\mathbf{l})$) is still learned only approximately. In our experiments, a model trained with this hybrid learning algorithm has the highest classification accuracy, beating both a generative model trained using CD as well as a purely discriminative model.

The main steps of the algorithm are listed below.

Hybrid learning algorithm for $P(\mathbf{v}, \mathbf{l})$:
Let $\{\mathbf{v}^+, l_k^+ = 1\}$ be a labeled training case.
**Generative update: CD learning of $P(\mathbf{v}|\mathbf{l})$**

1. Sample $\mathbf{h}^+ \sim P(\mathbf{h}|\mathbf{v}^+, l_k^+ = 1)$.
2. Compute the outer product $D_k^+ = \mathbf{v}^+\mathbf{h}^{+T}$.
3. Sample $\mathbf{v}^- \sim P(\mathbf{v}|\mathbf{h}^+, l_k^+ = 1)$.
4. Sample $\mathbf{h}^- \sim P(\mathbf{h}|\mathbf{v}^-, l_k^+ = 1)$.
5. Compute the outer product $D_k^- = v^- h^{-T}$.
6. Compute update $\Delta W_{\cdot,\cdot,k}^g = D_k^+ - D_k^-$.

**Discriminative update: ML learning of $P(\mathbf{l}|\mathbf{v})$**

1. Compute $\log P(l_c = 1|\mathbf{v}^+)$ for $c \in \{1, ..., N_l\}$.
2. Using the result from step 1 and the true label $l_k^+ = 1$, compute the update
   $\Delta W_{\cdot,\cdot,k}^d = \partial \log P(\mathbf{l}|\mathbf{v})/\partial W_{\cdot,\cdot,c}$ for $c \in \{1, ..., N_l\}$.

The two types of update for the $c^{th}$ slice of the tensor $W_{\cdot,\cdot,c}$ are then combined by a weighted sum:

$$W_{\cdot,\cdot,c} \leftarrow W_{\cdot,\cdot,c} + \eta(\Delta W_{\cdot,\cdot,c}^g + \lambda \Delta W_{\cdot,\cdot,c}^d), \tag{7}$$

where $\lambda$ is a parameter that sets the relative weighting of the generative and discriminative updates, and $\eta$ is the learning rate. As before, the updates from a mini-batch of training cases can be averaged together and applied as a single update to the parameters. In experiments, we set $\lambda$ by trying different values and evaluating classification accuracy on a validation set.

Note that the generative part in the above algorithm is simply CD learning of the RBM for the $k^{th}$ class. The earlier problem of slow mixing does not appear in the hybrid algorithm because the chain in the generative part does not involve sampling the label.

**Semi-supervised learning:** The hybrid learning algorithm can also make use of *unlabeled* training cases by treating their labels as missing inputs. The model first infers the missing label by sampling $P(\mathbf{l}|\mathbf{v}_u)$ for an unlabeled training case $\mathbf{v}_u$. The generative update is then computed by treating the inferred label as the true label. (The discriminative update will always be zero in this case.) Therefore the unlabeled training cases contribute an extra generative term to the parameter update.

## 4   Sparsity

Discriminative performance is improved by using binary features that are only rarely active. Sparse activities are achieved by specifying a desired probability of being active, $p << 1$, and then adding an additional penalty term that encourages an exponentially decaying average, $q$, of the actual probability of being active to be close to $p$. The natural error measure to use is the cross entropy between the desired and actual distributions: $p \log q + (1-p) \log(1-q)$. For logistic units this has a simple derivative of $p-q$ with respect to the total input to a unit. This derivative is used to adjust both the bias and the incoming weights of each hidden unit. We tried various values for $p$ and 0.1 worked well. In addition to specifying $p$ it is necessary to specify how fast the estimate of $q$ decays. We used $q_{new} = 0.9 * q_{old} + 0.1 * q_{current}$ where $q_{current}$ is the average probability of activation for the current mini-batch of 100 training cases. It is also necessary to specify how strong the penalty term should be, but this is easy to set empirically. We multiply the penalty gradient by a coefficient that is chosen to ensure that, on average, $q$ is close to $p$ but there is still significant variation among the $q$ values for different hidden units. This prevents the penalty term from dominating the learning. One

added advantage of this sparseness penalty is that it revives any hidden units whose average activities are much lower than $p$.

# 5  Evaluating DBNs on the NORB Object Recognition Task

## 5.1  NORB Database

For a detailed description see [12]. The five object classes in NORB are *animals*, *humans*, *planes*, *trucks*, and *cars*. The dataset comes in two different versions, *normalized-uniform* and *jittered-cluttered*. In this paper we use the *normalized-uniform* version, which has objects centred in the images with a uniform background. There are 10 instances of each object class, imaged under 6 illuminations and 162 viewpoints (18 azimuths × 9 elevations). The instances are split into two disjoint sets (pre-specified in the database) of five each to define the training and test sets, both containing 24,300 cases. So at test time a trained model has to recognize *unseen instances* of the same object classes.

**Pre-processing:** A single training (and test) case is a stereo-pair of grayscale images, each of size 96×96. To speed up experiments, we reduce dimensionality by using a "foveal" image representation. The central $64 \times 64$ portion of an image is kept at its original resolution. The remaining 16 pixel-wide ring around it is compressed by replacing non-overlapping square blocks of pixels with the average value of a block. We split the ring into four smaller ones: the outermost ring has $8 \times 8$ blocks, followed by a ring of $4 \times 4$ blocks, and finally two innermost rings of $2 \times 2$ blocks. The foveal representation reduces the dimensionality of a stereo-pair from 18432 to 8976. All our models treat the stereo-pair images as 8976-dimensional vectors[1].

## 5.2  Training Details

**Model architecture:** The two main decisions to make when training DBNs are the number of hidden layers to greedily pre-train and the number of hidden units to use in each layer. To simplify the experiments we constrain the number of hidden units to be the same at all layers (including the top-level model). We have tried hidden layer sizes of 2000, 4000, and 8000 units. We have also tried models with two, one, or no greedily pre-trained hidden layers. To avoid clutter, only the results for the best settings of these two parameters are given. The best classification results are given by the DBN with one greedily pre-trained sparse hidden layer of 4000 units (regardless of the type of top-level model).

A DBN trained on the pre-processed input with one greedily pre-trained layer of 4000 hidden units and a third-order model on top of it, also with 4000 hidden units, has roughly 116 million learnable parameters in total. This is roughly two orders of magnitude more parameters than some of the early DBNs trained on the MNIST images [6], [10]. Training such a model in Matlab on an Intel Xeon 3GHz machine takes almost two weeks. See a recent paper by Raina et al. [15] that uses GPUs to train a deep model with roughly the same number of parameters much more quickly.

We put Gaussian units at the lowest (pixel) layer of the DBN, which have been shown to be effective for modelling grayscale images [7]. See [7], [21] for details about Gaussian units.

# 6  Results

The results are presented in three parts: part 1 compares deep models to shallow ones, all trained using CD. Part 2 compares CD to the hybrid learning algorithm for training the top-level model of a DBN. Part 3 compares DBNs trained with and without unlabeled data, using either CD or the hybrid algorithm at the top level. For comparison, here are some published results for discriminative models on normalized-uniform NORB (without any pre-processing) [2], [12]: logistic regression 19.6%, kNN (k=1) 18.4%, Gaussian kernel SVM 11.6%, convolutional neural net 6.0%, convolutional net + SVM hybrid 5.9%.

## 6.1 Deep vs. Shallow Models Trained with CD

We consider here DBNs with one greedily pre-trained layer and a top-level model that contains the greedily pretrained features as its "visible" layer. The corresponding shallow version trains the top-level model directly on the pixels (using Gaussian visible units), with no pre-trained layers in between. Using CD as the learning algorithm (for both greedy pre-training and at the top-level) with the two types of top-level models gives us four possibilities to compare. The test error rates for these four models(see table 1) show that one greedily pre-trained layer reduces the error substantially, even without any subsequent fine-tuning of the pre-trained layer.

| Model | RBM with label unit | Third-order RBM |
|---------|:---:|:---:|
| Shallow | 22.8% | 20.8% |
| Deep | 11.9% | 7.6% |

Table 1: NORB test set error rates for deep and shallow models trained using CD with two types of top-level models.

The third-order RBM outperforms the standard RBM top-level model when they both have the *same number of hidden units*, but a better comparison might be to match the number of *parameters* by increasing the hidden layer size of the standard RBM model by five times (i.e. 20000 hidden units). We have tried training such an RBM, but the error rate is worse than the RBM with 4000 hidden units.

## 6.2 Hybrid vs. CD Learning for the Top-level Model

We now compare the two alternatives for training the top-level model of a DBN. There are four possible combinations of top-level models and learning algorithms, and table 2 lists their error rates. All these DBNs share the same greedily pre-trained first layer – only the top-level model differs among them.

| Learning algorithm | RBM with label unit | Third-order RBM |
|---------|:---:|:---:|
| CD | 11.9% | 7.6% |
| Hybrid | 10.4% | 6.5% |

Table 2: NORB test set error rates for top-level models trained using CD and the hybrid learning algorithms.

The lower error rates of hybrid learning are partly due to its ability to avoid the poor mixing of the label variable when CD is used to learn the joint density $P(\mathbf{v}, \mathbf{l})$ and partly due to its greater emphasis on discrimination (but with strong regularization provided by also learning $P(\mathbf{v}|\mathbf{l})$).

## 6.3 Semi-supervised vs. Supervised Learning

In this final part, we create additional images from the original NORB training set by applying global translations of 2, 4, and 6 pixels in eight directions (two horizontal, two vertical and four diagonal directions) to the original stereo-pair images[2]. These "jittered" images are treated as extra *unlabeled* training cases that are combined with the original labeled cases to form a much larger training set. Note that we could have assigned the jittered images the same class label as their source images. By treating them as unlabeled, the goal is to test whether improving the unsupervised, generative part of the learning alone can improve discriminative performance.

There are two ways to use unlabeled data:

1. Use it for greedy pre-training of the lower layers only, and then train the top-level model as before, with only labeled data and the hybrid algorithm.

2. Use it for learning the top-level model as well, now with the semi-supervised variant of the hybrid algorithm at the top-level.

Table 3 lists the results for both options.

| Top-level model (hyrbid learning only) | Unlabeled jitter for pre-training lower layer? | Unlabeled jitter at the top-level? | Error |
|---|---|---|---|
| RBM with label unit | No | No | 10.4% |
| | Yes | No | 9.0% |
| Third-order model | No | No | 6.5% |
| | Yes | No | 5.3% |
| | Yes | Yes | 5.2% |

Table 3: NORB test set error rates for DBNs trained with and without unlabeled data, and using the hybrid learning algorithm at the top-level.

The key conclusion from table 3 is that simply using more *unlabeled* training data in the unsupervised, greedy pre-training phase alone can significantly improve the classification accuracy of the DBN. It allows a third-order top-level model to reduce its error from 6.5% to 5.3%, which beats the current best published result for normalized-uniform NORB *without using any extra labeled data.* Using more unlabeled data also at the top level further improves accuracy, but only slightly, to 5.2%.

Now consider a discriminative model at the top, representing the distribution $P(\mathbf{l}|\mathbf{v})$. Unlike in the generative case, the exact gradient of the log-likelihood is tractable to compute. Table 4 shows the results of some discriminative models. These models use the same greedily pre-trained lower layer, learned with unlabeled jitter. They differ in how the top-level parameters are initialized, and whether they use the jittered images as extra labeled cases for learning $P(\mathbf{l}|\mathbf{v})$.

We compare training the discriminative top-level model "from scratch" (random initialization) versus initializing its parameters to those of a generative model learned by the hybrid algorithm. We also compare the effect of using the jittered images as extra *labeled* cases. As mentioned before, it is possible to assign the jittered images the same labels as the original NORB images they are generated from, which expands the labeled training set by 25 times. The bottom two rows of table 4 compare a discriminative third-order model initialized with and without pre-training. Pre-trained initialization (5.0%)

| Initialization of top-level parameters | Use jittered images as labeled? | Error |
|---|---|---|
| Random | No | 13.4% |
| Random | Yes | 7.1% |
| Model with 5.2% error from table 3 | Yes | 5.0% |

Table 4: NORB test set error rates for discriminative third-order models at the top level.

significantly improves accuracy over random initialization (7.1%). But note that discriminative training only makes a small additional improvement (5.2% to 5.0%) over the accuracy of the pre-trained model itself.

## 7 Conclusions

Our results make a strong case for the use of generative modeling in object recognition. The main two points are: 1) Unsupervised, greedy, generative learning can extract an image representation that supports more accurate object recognition than the raw pixel representation. 2) Including $P(\mathbf{v}|\mathbf{l})$ in the objective function for training the top-level model results in better classification accuracy than using $P(\mathbf{l}|\mathbf{v})$ alone. In future work we plan to factorize the third-order Boltzmann machine as described in [18] so that some of the top-level features can be shared across classes.

## Footnotes

[1]Knowledge about image topology is used only along the (mostly empty) borders, and not in the central portion that actually contains the object.

[2]The same translation is applied to both images in the stereo-pair.

# References

[1] Y. Bengio, P. Lamblin, P. Popovici, and H. Larochelle. Greedy Layer-Wise Training of Deep Networks. In *NIPS*, 2006.

[2] Y. Bengio and Y. LeCun. Scaling learning algorithms towards AI. In *Large-Scale Kernel Machines*, 2007.

[3] D. DeCoste and B. Scholkopf. Training Invariant Support Vector Machines. *Machine Learning*, 46:161–190, 2002.

[4] G. E. Hinton. Training products of experts by minimizing contrastive divergence. *Neural Computation*, 14(8):1711–1800, 2002.

[5] G. E. Hinton. To Recognize Shapes, First Learn to Generate Images. *Technical Report UTML TR 2006-04, Dept. of Computer Science, University of Toronto*, 2006.

[6] G. E. Hinton, S. Osindero, and Y. Teh. A fast learning algorithm for deep belief nets. *Neural Computation*, 18:1527–1554, 2006.

[7] G. E. Hinton and R. Salakhutdinov. Reducing the dimensionality of data with neural networks. *Science*, 313:504–507, 2006.

[8] M. Kelm, C. Pal, and A. McCallum. Combining Generative and Discriminative Methods for Pixel Classification with Multi-Conditional Learning. In *ICPR*, 2006.

[9] H. Larochelle and Y. Bengio. Classification Using Discriminative Restricted Boltzmann Machines. In *ICML*, pages 536–543, 2008.

[10] H. Larochelle, D. Erhan, A. Courville, J. Bergstra, and Y. Bengio. An empirical evaluation of deep architectures on problems with many factors of variation. In *ICML*, pages 473–480, 2007.

[11] Y. LeCun, L. Bottou, Y. Bengio, and P. Haffner. Gradient-based learning applied to document recognition. *Proceedings of the IEEE*, 86(11):2278–2324, November 1998.

[12] Y. LeCun, F. J. Huang, and L. Bottou. Learning methods for generic object recognition with invariance to pose and lighting. In *CVPR*, Washington, D.C., 2004.

[13] H. Lee, R. Grosse, R. Ranganath, and A. Ng. Convolutional Deep Belief Networks for Scalable Unsupervised Learning of Hierarchical Representations. In *ICML*, 2009.

[14] V. Nair and G. E. Hinton. Implicit mixtures of restricted boltzmann machines. In *Neural information processing systems*, 2008.

[15] R. Raina, A. Madhavan, and A. Ng. Large-scale Deep Unsupervised Learning using Graphics Processors. In *ICML*, 2009.

[16] Marc'Aurelio Ranzato, Fu-Jie Huang, Y-Lan Boureau, and Yann LeCun. Unsupervised learning of invariant feature hierarchies with applications to object recognition. In *Proc. Computer Vision and Pattern Recognition Conference (CVPR'07)*. IEEE Press, 2007.

[17] T. J. Sejnowski. Higher-order Boltzmann Machines. In *AIP Conference Proceedings*, pages 398–403, 1987.

[18] G. Taylor and G. E. Hinton. Factored Conditional Restricted Boltzmann Machines for Modeling Motion Style. In *ICML*, 2009.

[19] V. Vapnik. *Statistical Learning Theory*. John Wiley and Sons, 1998.

[20] P. Vincent, H. Larochelle, Y. Bengio, and P. A. Manzagol. Extracting and Composing Robust Features with Denoising Autoencoders. In *ICML*, 2008.

[21] M. Welling, M. Rosen-Zvi, and G. E. Hinton. Exponential family harmoniums with an application to information retrieval. In *NIPS 17*, 2005.

